# A Neural Network Based Head Tracking System

**D. D. Lee and H. S. Seung**
Bell Laboratories, Lucent Technologies
700 Mountain Ave.
Murray Hill, NJ 07974
{ddlee|seung}@bell-labs.com

## Abstract

We have constructed an inexpensive, video-based, motorized tracking system that learns to track a head. It uses real time graphical user inputs or an auxiliary infrared detector as supervisory signals to train a convolutional neural network. The inputs to the neural network consist of normalized luminance and chrominance images and motion information from frame differences. Subsampled images are also used to provide scale invariance. During the online training phase, the neural network rapidly adjusts the input weights depending upon the reliability of the different channels in the surrounding environment. This quick adaptation allows the system to robustly track a head even when other objects are moving within a cluttered background.

## 1 Introduction

With the proliferation of inexpensive multimedia computers and peripheral equipment, video conferencing finally appears ready to enter the mainstream. But personal video conferencing systems typically use a stationary camera, tying the user to a fixed location much as a corded telephone tethers one to the telephone jack. A simple solution to this problem is to use a motorized video camera that can track a specific person as he or she moves about. However, this presents the difficulty of having to continually control the movements of the camera while one is communicating. In this paper, we present a prototype, neural network based system that learns the characteristics of a person's head in real time and automatically tracks it around the room, thus alleviating the user of much of this burden.

The camera movements in this video conferencing system closely resemble the movements of human eyes. The task of the biological oculomotor system is to direct

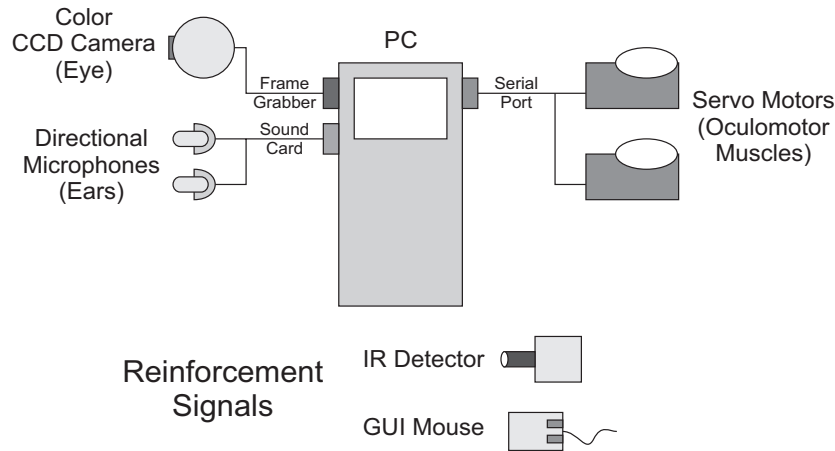

Figure 1: Schematic hardware diagram of Marvin, our head tracking system.

"interesting" parts of the visual world onto the small, high resolution areas of the retinas. For this task, complex neural circuits have evolved in order to control the eye movements. Some examples include the saccadic and smooth pursuit systems that allow the eyes to rapidly acquire and track moving objects [1, 2]. Similarly, an active video conferencing system also needs to determine the appropriate face or feature to follow in the video stream. Then the camera must track that person's movements over time and transmit the image to the other party.

In the past few years, the problem of face detection in images and video has attracted considerable attention [3, 4, 5]. Rule-based methods have concentrated on looking for generic characteristics of faces such as oval shapes or skin hue. Since these types of algorithms are fairly simple to implement, they are commonly found in real-time systems [6, 7]. But because other objects have similar shapes and colors as faces, these systems can also be easily fooled. A potentially more robust approach is to use a convolutional neural network to learn the appropriate features of a face [8, 9]. Because most such implementations learn in batch mode, they are beset by the difficulty of constructing a large enough training set of labelled images with and without faces. In this paper, we present a video based system that uses online supervisory signals to train a convolutional neural network. Fast online adaptation of the network's weights allows the neural network to learn how to discriminate an individual head at the beginning of a session. This enables the system to robustly track the head even in the presence of other moving objects.

## 2   Hardware Implementation

Figure 1 shows a schematic of the tracking system we have constructed and have named "Marvin" because of an early version's similarity to a cartoon character. Marvin's eye consists of a small CCD camera with a 65° field of view that is attached to a motorized platform. Two RC servo motors give Marvin the ability to rapidly pan and tilt over a wide range of viewing angles, with a typical maximum velocity of 300 deg/sec. The system also includes two microphones or ears that give Marvin the ability to locate auditory cues. Integrating auditory information with visual inputs allows the system to find salient objects better than with either sound or video alone. But these proceedings will focus exclusively on how a visual representation is learned.

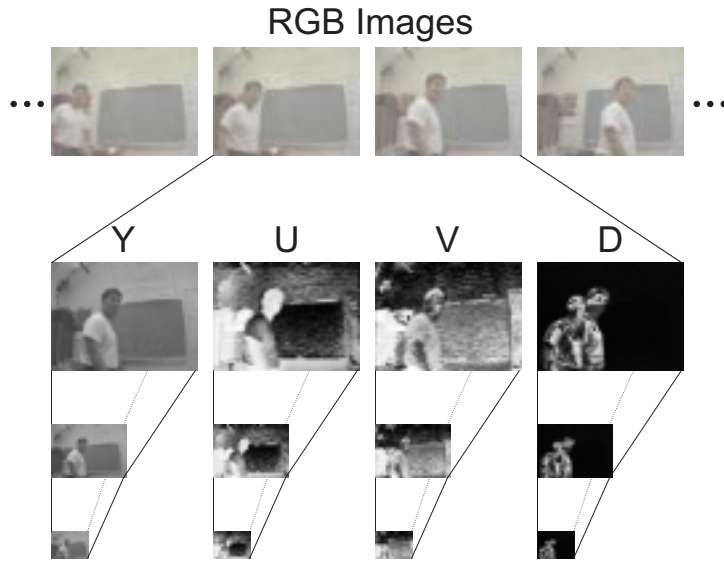

Figure 2: Preprocessing of the video stream. Luminance, chromatic and motion information are separately represented in the Y, U, V, D channels at multiple resolutions.

Marvin is able to learn to track a visual target using two different sources of supervisory signals. One method of training uses a small 38 KHz modulated infrared light emitter ($\lambda \approx 900\,\mathrm{nm}$) attached to the object that needs to be tracked. A heat filter renders the infrared light invisible to Marvin's video camera so that the system does not merely learn to follow this signal. But mounted next to the CCD camera and moving with it is a small infrared detector with a collimating lens that signals when the object is located within a narrow angular cone in the direction that the camera is pointing. This reinforcement signal can then be used to train the weights of the neural network. Another more natural way for the system to learn occurs in an actual video conferencing scenario. In this situation, a user who is actively watching the video stream has manual override control of the camera using graphical user interface inputs. Whenever the user repositions the camera to a new location, the neural network would then adjust its weights to track whatever is in the center portion of the image.

Since Marvin was built from readily available commercial components, the cost of the system not including the PC was under $500. The input devices and motors are all controlled by the computer using custom-written Matlab drivers that are available for both Microsoft Windows and the Linux operating system. The image processing computations as well as the graphical user interface are then easily implemented as simple Matlab operations and function calls. The following section describes the head tracking neural network in more detail.

## 3   Neural Network Architecture

Marvin uses a convolutional neural network architecture to detect a head within its field of view. The video stream from the CCD camera is first digitized with a video capture board into a series of raw $120 \times 160$ RGB images as shown in Figure 2. Each RGB color image is then converted into its YUV representation, and a difference (D)

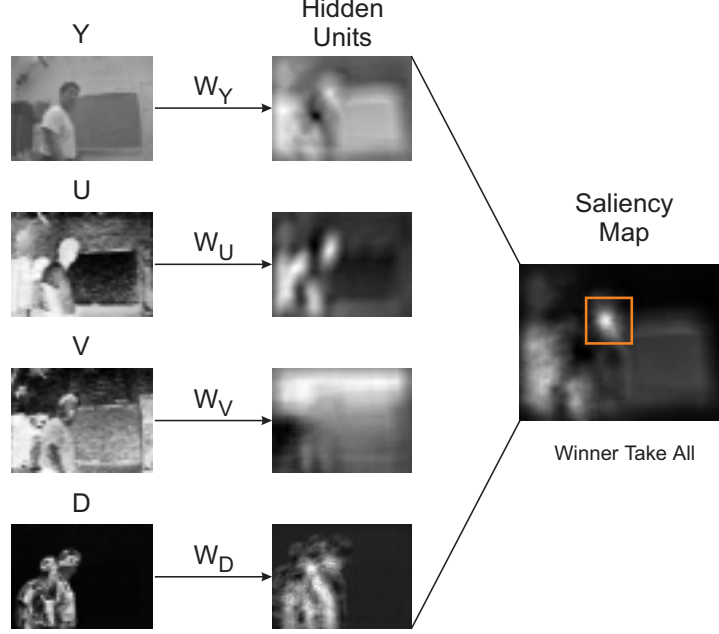

Figure 3: Neural network uses a convolutional architecture to integrate the different sources of information and determine the maximally salient object.

image is also computed as the absolute value of the difference from the preceding frame. Of the four resulting images, the Y component represents the luminance or grayscale information while the U and V channels contain the chromatic or color information. Motion information in the video stream is captured by the D image where moving objects appear highlighted.

The four YUVD channels are then subsampled successively to yield representations at lower and lower resolutions. The resulting "image pyramids" allow the network to achieve recognition invariance across many different scales without having to train separate neural networks for each resolution. Instead, a single neural network with the same set weights is run with the different resolutions as inputs, and the maximally active resolution and position is selected.

Marvin uses the convolutional neural network architecture shown in Figure 3 to locate salient objects at the different resolutions. The YUVD input images are filtered with separate $16 \times 16$ kernels, denoted by $W_Y$, $W_U$, $W_V$, and $W_D$ respectively. This results in the filtered images $\bar{Y}^s$, $\bar{U}^s$, $\bar{V}^s$, $\bar{D}^s$:

$$\bar{A}^s(i,j) = W_A \circ A^s = \sum_{i',j'} W_A(i',j') A^s(i+i',j+j') \tag{1}$$

where $s$ denotes the scale resolution of the inputs, and $A$ is any of the $Y$, $U$, $V$, or $D$ channels. These filtered images represent a single layer of hidden units in the neural network. These hidden units are then combined to form the saliency map $X^s$ in the following manner:

$$X^s(i,j) = c_Y \, g[\bar{Y}^s(i,j)] + c_U \, g[\bar{U}^s(i,j)] + c_V \, g[\bar{V}^s(i,j)] + c_D \, g[\bar{D}^s(i,j)] + c_0. \tag{2}$$

Since $g(x) = \tanh(x)$ is sigmoidal, the saliency $X^s$ is computed as a nonlinear, pixel-by-pixel combination of the hidden units. The scalar variables $c_Y$, $c_U$, $c_V$, and $c_D$ represent the relative importance of the different luminance, chromatic, and motion channels in the overall saliency of an object.

With the bias term $c_0$, the function $g[X^s(i,j)]$ may then be thought of as the relative probability that a head exists at location $(i,j)$ at input resolution $s$. The final output of the neural network is then determined in a competitive manner by finding the location $(i_m, j_m)$ and scale $s_m$ of the best possible match:

$$g[X_m] = g[X^{s_m}(i_m, j_m)] = \max_{i,j,s} g[X^s(i,j)]. \tag{3}$$

After processing the visual inputs in this manner, saccadic camera movements are generated in order to keep the maximally salient object located near the center of the field of view.

## 4   Training and Results

Either GUI user inputs or the infrared detector may be used as a supervisory signal to train the kernels $W_A$ and scalar weights $c_A$ of the neural network. The neural network is updated when the maximally salient location of the neural network $(i_m, j_m)$ does not correspond to the desired object's true position $(i_n, j_n)$ as identified by the external supervisory signal. A cost function proportional to the sum squared error terms at the maximal location and new desired location is used for training:

$$e_m^2 = |g_m - g[X^{s_m}(i_m, j_m)]|^2, \tag{4}$$

$$e_n^2 = \min_s |g_n - g[X^s(i_n, j_n)]|^2. \tag{5}$$

In the following examples, the constants $g_m = 0$ and $g_n = 1$ are used. The gradients to Eqs. 4–5 are then backpropagated through the convolutional network [8, 10], resulting in the following update rules:

$$\Delta c_A = \eta\, e_m g'(X_m) g[\bar{A}(i_m, j_m)] + \eta\, e_n g'(X_n) g[\bar{A}(i_n, j_n)], \tag{6}$$

$$\Delta W_A = \eta\, e_m g'(X_m) g'(\bar{A}_m) c_A A_m + \eta\, e_n g'(X_n) g'(\bar{A}_n) c_A A_n. \tag{7}$$

In typical batch learning applications of neural networks, the learning rate $\eta$ is set to be some small positive number. However in this case, it is desirable for Marvin to learn to track a head in a new environment as quickly as possible. Thus, rapid adaptation of the weights during even a single training example is needed. A natural way of doing this is to use a fairly large learning rate ($\eta = 0.1$), and to repeatedly apply the update rules in Eqs. 6–7 until the calculated maximally salient location is very close to the actual desired position.

An example of how quickly Marvin is able to learn to track one of the authors as he moved around his office is given by the learning curve in Figure 4. The weights were first initialized to small random values, and Marvin was corrected in an online fashion using mouse inputs to look at the author's head. After only a few seconds of training with a processing time loop of around 200 ms, the system was able to locate the head to within four pixels of accuracy, as determined by hand labelling the video data afterwards. As saccadic eye movements were initiated at

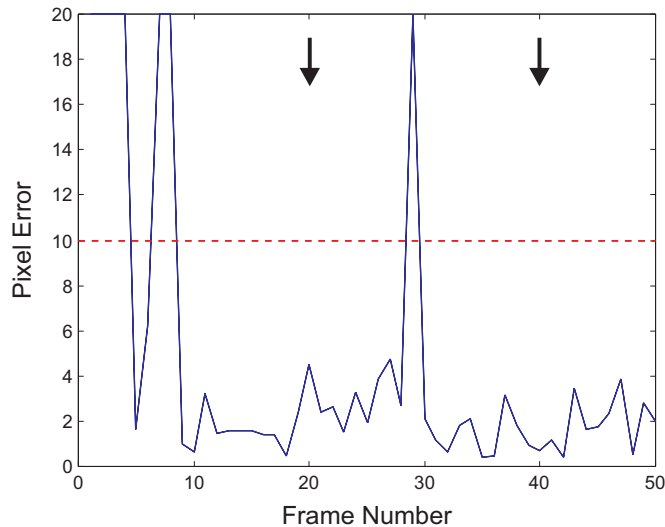

Figure 4: Fast online adaptation of the neural network. The head location error in pixels in a $120 \times 160$ image is plotted as a function of frame number (5 frames/sec).

the times indicated by the arrows in Fig. 4, new environments of the office were sampled and an occasional large error is seen. However, over time as these errors are corrected, the neural network learns to robustly discriminate the head from the office surroundings.

# 5    Discussion

Figure 5 shows the inputs and weights of the network after a minute of training as the author walked around his office. The kernels necessarily appear a little smeared because they are invariant to slight changes in head position, rotation, and scale. But they clearly depict the dark hair, facial features, and skin color of the head. The relative weighting ($c_Y, c_U, c_V > c_D$) of the different input channels shows that the luminance and color information are the most reliable for tracking the head. This is probably because it is relatively difficult to distinguish in the frame difference images the head from other moving body parts.

We are currently considering more complicated neural network architectures for combining the different input streams to give better tracking performance. However, this example shows how a simple convolutional architecture can be used to automatically integrate different visual cues to robustly track a head. Moreover, by using fast online adaptation of the neural network weights, the system is able to learn without needing large hand-labelled training sets and is also able to rapidly accomodate changing environments. Future improvements in hardware and neural network architectures and algorithms are still necessary, however, in order to approach human speeds and performance in this type of sensory processing and recognition task.

We acknowledge the support of Bell Laboratories, Lucent Technologies. We also thank M. Fee, A. Jacquin, S. Levinson, E. Petajan, G. Pingali, and E. Rietman for helpful discussions.

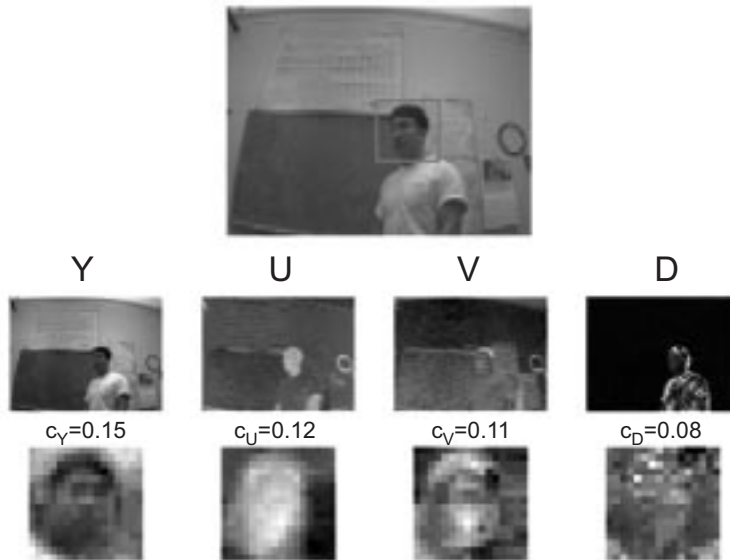

Figure 5: Example showing the inputs and weights used in tracking a head. The head position as calculated by the neural network is marked with a box.

## References

[1] Horiuchi, TK, Bishofberger, B & Koch, C (1994). An analog VLSI saccadic eye movement system. *Advances in Neural Information Processing Systems 6*, 582–589.

[2] Rao, RPN, Zelinsky, GJ, Hayhoe, MM & Ballard, DH (1996). Modeling saccadic targeting in visual search. *Advances in Neural Information Processing Systems 8*, 830–836.

[3] Sung, KK & Poggio, T (1994). Example-based learning for view-based human face detection. *Proc. 23rd Image Understanding Workshop*, 843–850.

[4] Eleftheriadis, A & Jacquin, A (1995). Automatic face location detection and tracking for model-assisted coding of video teleconferencing sequences at low bit-rates. *Signal Processing: Image Communication* **7**, 231.

[5] Petajan, E & Graf, HP (1996). Robust face feature analysis for automatic speechreading and character animation. *Proc. 2nd Int. Conf. Automatic Face and Gesture Recognition*, 357-362.

[6] Darrell, T, Maes, P, Blumberg, B, & Pentland, AP (1994). A novel environment for situated vision and behavior. *Proc. IEEE Workshop for Visual Behaviors*, 68–72.

[7] Yang, J & Waibel, A (1996). A real-time face tracker. *Proc. 3rd IEEE Workshop on Application of Computer Vision*, 142–147.

[8] Nowlan, SJ & Platt, JC (1995). A convolutional neural network hand tracker. *Advances in Neural Information Processing Systems 7*, 901–908.

[9] Rowley, HA, Baluja, S & Kanade, T (1996). Human face detection in visual scenes. *Advances in Neural Information Processing Systems 8*, 875–881.

[10] Le Cun, Y, et al. (1990). Handwritten digit recognition with a back propagation network. *Advances in Neural Information Processing Systems 2*, 396–404.
